# Rethinking Memory and Communication Costs for Efficient Data Parallel Training of Large Language Models

**Hanxiao Zhang, Lin Ju, Chan Wu, Jinjing Huang, Youshao Xiao**∗
**Zhenglei Zhou**, **Zhiming Fan**, **Zhaoxin Huan**, **Siyuan Li**, **Fanzhuang Meng**
**Lei Liang**, **Xiaolu Zhang**, **Jun Zhou**∗
Ant Group, Hangzhou, China
{zhanghanxiao.zhx,julin.jl,wuchan.wu,huangjinjing.hjj,youshao.xys,
zhouzhenglei.zzl,zhiming.fzm,zhaoxin.hzx,lisiyuan.li,mengfanzhuang.mfz,
leywar.liang,yueyin.zxl,jun.zhoujun}@antgroup.com

## Abstract

Recently, various strategies for distributed training of large language models (LLMs) have been proposed. By categorizing them into basic strategies and composite strategies, we have discovered that existing basic strategies provide limited options in specific scenarios, leaving considerable room for optimization in training speed. In this paper, we rethink the impact of memory and communication costs on the training speed of LLMs when employ data parallelism based techniques. We take the impact of intra- and inter-group communication performance disparities into account , and then propose a new set of basic strategies named the **Pa**rtial **R**edundancy **O**ptimizer (PaRO). PaRO Data Parallelism (PaRO-DP) accelerates LLM training through refined model state partitioning and tailored training procedures. Additionally, PaRO Collective Communications (PaRO-CC) speeds up collective communication operations by rearranging the topology. We also propose a guideline for choosing different DP strategies based on simple quantitative calculations, which yields minimal ranking errors. Our experiments show that PaRO improves the training speed of LLMs by up to 266% that of ZeRO-3 as basic DP strategies. Moreover, employing PaRO-CC independently for model parallel strategies, such as Megatron, can also boost the training speed by 17%.

## 1 Introduction

Large language models (LLMs) have demonstrated extraordinary capabilities across various domains, expanding their parameter sizes into the tens or even hundreds of billions [3, 4, 29]. To tackle the intricate challenge of training such large models, different distributed training strategies have been proposed.

Data Parallelism (DP) [1, 18, 34] divides the input data into multiple mini-batches, with each GPU independently processing a mini-batch through the forward and backward passes, followed by synchronization of the gradients. ZeRO [25] as a DP strategy, suggests partitioning model states across multiple GPUs. This strategy facilitates a trade-off between memory and communication costs, offering three distinct partitioning methods, known as ZeRO-1, ZeRO-2 and ZeRO-3.

Model Parallelism (MP) [23, 19], which encompasses Tensor Parallelism (TP) [27, 35, 31, 2] and Pipeline Parallelism (PP) [13, 10], distributes the model components across multiple GPUs. For instance, Megatron-LM [27] achieves 1D tensor parallelism by distributing the row or column

---

∗Corresponding Authors

dimensions of the Transformer layers across multiple GPUs. However, the output of each layer requires all-reduce operations to aggregate the input, which increases the cost of communication.

We classify these strategies as **basic strategies** for distributed training. We argue that *choosing a basic strategy essentially involves making a trade-off between memory and communication costs, based on different model sizes and hardware configurations to achieve a faster training speed.*

Beyond the traditional approach of training on the full set of model parameters, Low-Rank Adaptation (LoRA) [12] is frequently employed in the fine-tuning of large models as a form of Parameter-Efficient Fine-Tuning (PEFT) [21] methods. Present approaches to speed up PEFT, such as QLORA [7] and LoRA-FA [38], leverage quantization and flash-attention [6] techniques. However, there is a lack of discussion on the distributed training strategy for finetuning scenarios including PEFT.

Additionally, mainstream GPU clusters adopt a heterogeneous network architecture that utilizes NVLink within machines and Remote Direct Memory Access (RDMA) between machines. Similarly, a set of GPU machines are interconnected via a high-performance switch, whereas a training cluster comprises multiple sets of GPU machines that need to communicate across switches [9]. In this paper, we refer to a multi-GPU machine or a switch as a **group**, applicable to other heterogeneous network scenarios as well. The constraint of inter-group communication performance being poorer than intra-group communication performance can slow down the collective communication operations that are commonly used in DP and MP methods. Moreover, for methods like ZeRO, the entire training process needs to be redesigned because there is a considerable amount of communication operations during the forward pass, backward pass, and gradient update stages.

In the analysis of various mainstream basic distributed training strategies, we consider two key questions:

- *Are there more optimal basic strategies that offer additional trade-off options for the DP training, including suitable strategies for scenarios involving full-parameters training and partial-parameters training, thereby accelerating the training speed?*

- *Is there a more optimal collective communication method that can provide more efficient communication for DP and MP methods, thereby accelerating the training speed?*

This paper addresses these questions by exploring potential basic strategies for LLM distributed training. The main contributions of this paper are as follows:

- We systematically take into account the disparity in intra- and inter-group communication performance when rethinking the trade-off between memory and communication costs. Our in-depth analysis covers various scenarios involving both full-parameter training, partial-parameter training and PEFT.

- We propose the PaRO-DP set of strategies by refined model state partitioning and tailored training procedures. These strategies provide more options for the trade-off between memory and communication costs, allowing for training speed improvements of up to 266% over ZeRO in scenarios where inter-group communication performance is poor. Notably, our proposal of a tailored training strategy for PEFT constitutes a new attempt within the field to our knowledge.

- We propose a guideline that can be used to quantitatively select specific PaRO-DP strategies in various scenarios where the model size and communication performance vary. The effectiveness of this guideline has been validated via our extensive experiments.

- We propose PaRO Collective Communication (PaRO-CC) tailored for collective communication operations on clusters with performance disparities between intra- and inter-group communications. PaRO-CC can be incorporated into most basic training strategies, as demonstrated by a 17% improvement in a Megatron end-to-end training task. Compared to the traditional Ring topology, PaRO-CC exhibits a 36.5% improvement in collective communication efficiency.

Furthermore, we provide the open-source release of our code at `https://github.com/HanxiaoZhang/PaRO/tree/paro`.

## 2 Related Work

**Enhancement of Basic Strategies**  To enhance communication efficiency when training LLMs on large-scale GPU clusters, especially with heterogeneous networks, several methods have been proposed to employ group partitioning techniques to improve the basic strategy. ZeRO++[32] extends ZeRO-3 by performing a secondary partitioning of parameters while keeping other model states partitioned across all GPUs. This modification shifts the global all-gather operation in the backward pass to an intra-group all-gather, reducing the volume of inter-group communication. Additionally, ZeRO++ compresses model parameters and gradients using quantization to decrease communication volume. MiCS[39] introduces a group partitioning strategy, whereby the GPU cluster is divided into smaller and location-based subgroups. The model state is then partitioned within and replicated across these subgroups. By configuring appropriate subgroup sizes, MiCS can leverage high-quality intra-group networks and a hierarchical communication strategy to reduce communication volume between groups. PyTorch's official Fully Sharded Data Parallel (FSDP) provides a *HYBRID_SHARD* (FSDP-hs)[40] strategy that leverages data center locality like MiCS to accelerate training and reduce inter-group communication. These optimization approaches, which incorporate grouping strategies, have inspired the direction of our subsequent research efforts.

**Composite Strategy**  Combining multiple basic parallel strategies to integrate their benefits is a common practice. These combinations include multi-dimensional hybrid parallelism and auto-parallelism. Multi-dimensional hybrid parallelism refers to the combination of multiple parallel strategies for computation simultaneously in distributed training. By combining these parallel strategies judiciously, we can fully exploit the advantages of various resources to improve training efficiency. This method is widely used in the industry for training large language models, such as LLama[29], GLM[8], and Megatron-Turing NLG[28].

Auto-parallelism automatically selects a better or optimal parallel strategy for efficient execution based on the given model and machine resources used. Unlike hybrid parallel methods, automatic parallelism typically selects different basic strategies at a finer granularity, such as choosing different basic strategies for different layers or operators of the model. There are two types of auto-parallelism: semi-automatic and fully automatic. In semi-automatic mode, certain tensor and operator slices need to be specified, such as Mesh-TensorFlow[26], GShard[17], and GSPMD[36]. In fully automatic mode, the framework adaptively selects all tensors and operators to determine the best slicing strategy, including Alpa and Unity  [15, 41, 30, 22].

We argue that *through dedicated design efforts, new basic strategies can be leveraged across a spectrum of composite strategies*. This paper concentrates on the enhancement of basic strategies and refrains from comparing them directly with composite strategies.

## 3 PaRO Design

In the following discussion, we first delve into an analysis of the trade-off between memory and communication costs in LLM training. Based on this, we propose the PaRO-DP solution offering refined model state partitioning and a guideline for selecting an appropriate strategy in various scenarios. Additionally, we introduce PaRO-CC to further optimize the collective communication operation. Lastly, we discuss the applications of PaRO.

### 3.1 PaRO-DP

#### 3.1.1 Analysis and Insights

As aforementioned, a substantial performance gap of communication exists between intra- and inter-group networks, creating a bottleneck that impedes training efficiency. However, in data-parallel training tasks, by allowing for an acceptable level of memory redundancy, we can reduce the need for inter-group communication during the training process while fully leveraging high-performance communication within groups, thus reducing the overall communication cost.

Contrasting with ZeRO, our approach augments the model states with an additional intra-group partitioning state. For further illustration, we define three partitioning states: *No partitioning* ($N$), *Intra-group partitioning* ($I$) and *Global partitioning* ($G$) from coarse-grained to fine-grained, which

act on three components of model states: *parameter* ($p$), *gradient* ($g$), *optimizer state* ($os$). More specifically, $I$ means that model states are partitioned within the group, while each group holds a complete replica. $N$ means that each GPU holds a replica of model states. $G$ means that each GPU holds a part of model states, with only one complete model state on the global scale. We introduce the following notations to aid in the explanation. Note that, for simplification, we substitute network performance with bandwidth here.

| | | | |
|---|---|---|---|
| $m$: | The number of GPUs in the group. | $n$: | The number of GPUs in total. |
| $ng$: | The number of groups, $ng = n/m$. | $\Psi$: | The number of parameters. |
| $B$: | Inter-group bandwidth between GPUs. | $\Psi'$: | The number of trainable parameters. |
| $B'$: | Intra-group bandwidth between GPUs. | $s$: | The number of gradient accumulation. |
| $T$: | The Time cost of training a mini-batch. | | |
| $P_{p+g+os}$: | Partitioning strategy of $p$, $g$ and $os$. The value combines $N/I/G$ for each component of model states. For example, $P_{NIG}$ means the strategy with no partitioning of Parameter, intra-group partitioning of Gradient and global partitioning of Optimizer State. | | |

In the context of gradient accumulation where one mini-batch step contains $s$ micro-batch steps, we analyze the communication cost of model states with different partitioning states.

- **Parameter partitioning:** Parameters are utilized in both forward and backward computations during each step of the micro-batch. In both *global partitioning* and *intra-group partitioning* states, an all-gather operation is necessary to obtain all parameters of the current layer before use. While only an intra-group all-gather is required when partitioning model parameters within a group.
- **Gradient partitioning:** Gradients are computed during the backward computation and used in the model update stage. Likewise, in both *global partitioning* and *intra-group partitioning* states, the aggregated gradient of the corresponding local partition is obtained through the reduce-scatter operation. Only intra-group reduce-scatter is required when partitioning gradients within a group, which is quicker than the global reduce-scatter used in *global partitioning*. Subsequently, each GPU performs local accumulation of the aggregated gradients in every micro-batch step.
- **Optimizer state partitioning:** The optimizer state is utilized during the model updating stage. If the partitioning scope of optimizer states differs from that of gradients or parameters, communication operations will be needed before and after the optimizer step. For instance, it requires performing an inter-group reduce-scatter of gradients before the optimizer step and an inter-group all-gather of updated parameters after the optimizer step, when the partitioning strategy is $P_{IIG}$.

The above three levels of partitioning granularity on $p$, $g$ and $os$ bring up 27 combinations of model-partitioning strategies, while not all strategies are effective. Employing fine-grained partitioning can enhance memory efficiency for larger batch sizes, potentially increasing throughput, yet it also incurs more communication costs, thereby reducing throughput. Therefore, a trade-off between memory and communication costs should be made when selecting the appropriate model partitioning strategy.

In a mainstream training process using mixed precision and Adam optimizer[16], the memory consumption for the parameters, gradients and optimizer states are respectively $2\Psi$, $2\Psi'$, and $12\Psi'$ [25]. Note that in PEFT tasks, where $\Psi' \ll \Psi$, the sizes of $g$ and $os$ are relatively small compared to $p$. This suggests that PEFT tasks should be treated differently when making the trade-off between memory and communication costs.

In our findings, we identify a **key insight**: *in all scenarios involving various numbers of trainable parameters, the partitioning granularity of $os$ should be the same as or finer than that of $p$ and $g$*. Otherwise, if $os$ is partitioned by a coarser granularity than $p$ or $g$, more memory will be used without any reduction in communication cost, compared to when $os$ is partitioned by the same granularity of $p$ and $g$. According to this insight, we can eliminate 13 out of the 27 possible combinations mentioned earlier and Appendix A.1.1 provides more details.

### 3.1.2 Implementations of PaRO-DP

Following the analysis and resulting insights, we have formulated a strategy set of 14 combinations that constitute our proposed PaRO-DP. Specifically, DDP, ZeRO-1, ZeRO-2, ZeRO-3/FSDP, and

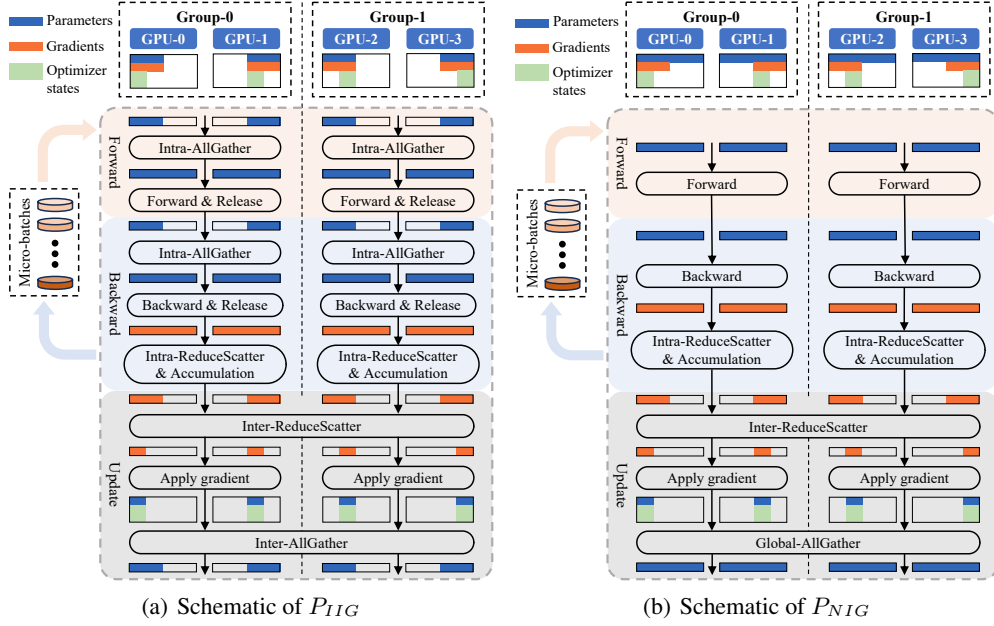

(a) Schematic of $P_{IIG}$          (b) Schematic of $P_{NIG}$

Figure 1: The schematic of PaRO-DP is illustrated within a cluster comprising four GPUs. Each group in the cluster consists of two GPU devices. In particular, we illustrate the computation and communication phases of a global step that incorporates gradient accumulation. We detail the intra- and inter-group communication by using specific prefixes for each collective communication primitive.

MiCS are equivalent or approximately equivalent to $P_{NNN}$, $P_{NNG}$, $P_{NGG}$, $P_{GGG}$, $P_{III}$ in the PaRO-DP strategy set.

As previously mentioned, when the model states are re-partitioned in different ways, the procedures for Forward, Backward, and Gradient Accumulation and Apply Gradient during the training process require redesigning. Below, we provide a detailed explanation of their training processes using $P_{IIG}$ and $P_{NIG}$ as examples. To simplify the examples, we use four GPUs and divide them into two groups to demonstrate the training process. These approaches can be easily scaled to larger GPU clusters and Appendix A.1 provides other solutions in detail. It includes explanations of various collective communication operations used in the diagrams.

Figure 1(a) illustrates the schematic of $P_{IIG}$, where the model parameters and gradients are intra-group partitioned, while the optimizer states are globally partitioned. Therefore, a complete replica of model parameters and gradients are preserved within each group. The memory cost of $P_{IIG}$ is less than that of $P_{NNG}$ (ZeRO-1), greater than $P_{GGG}$ (ZeRO-3), and in most scenarios less than $P_{NGG}$ (ZeRO-2), depending on the grouping situation. In the Forward and Backward stages, each GPU gathers a complete replica of model parameters through intra-group all-gather operations. After the backward computation, each GPU aggregates gradients from other GPUs through intra-group reduce-scatter operations for local gradient synchronization. These gradients are temporarily stored in each GPU group through gradient accumulation. Once the gradients of the last micro-batch are accumulated, each GPU performs inter-group reduce-scatter operations to achieve global gradient synchronization. Each GPU utilizes the gradient partition to update the optimizer state maintained by itself and update model parameters. Finally, the updated model parameters of each GPU are obtained from other groups through inter-group all-gather operations. Note that compared to ZeRO-3, $P_{IIG}$ employs intra-group collective communication in place of global collective communication in the Forward and Backward stages of each micro-batch step. This makes $P_{IIG}$ faster than ZeRO-3 in our context, at the cost of additional but limited memory of parameters and gradients.

Figure 1(b) illustrates the schematic of $P_{NIG}$, where the parameters of the model are not partitioned, while the gradients are intra-group partitioned, and the optimizer states are globally partitioned. The memory cost of $P_{NIG}$ is less than that of $P_{NNG}$ (ZeRO-1), greater than $P_{NGG}$ (ZeRO-2). Different

from $P_{IIG}$, each GPU retains complete model parameters in $P_{NIG}$. Therefore, in the Forward and Backward stages, each GPU can directly perform the computation without collecting and releasing model parameters. The subsequent four-step computation process of $P_{NIG}$ is consistent with that of $P_{IIG}$. Finally, each GPU collects updated parameters via PaRO-CC all-gather operations. Note that compared to ZeRO-2, $P_{NIG}$ employs intra-group collective communication in place of global collective communication during the gradient accumulation of each micro-batch step. This makes $P_{NIG}$ faster than ZeRO-2 in our context, at the cost of additional but limited memory redundancy of gradients.

Moreover, PaRO-DP strategies show efficiency advantages over both ZeRO++ and FSDP_HYBRID_SHARD_ZERO2 (FSDP-hsz) approaches. Notably, under equal peak memory conditions, the global all-gather operations in the Forward stage of ZeRO++ are less efficient than $P_{IGG}$'s intra-group all-gather. Similarly, FSDP-hsz incurs additional communication costs due to the all-gather operations of parameters in the Forward stage compared to $P_{NII}$.

### 3.1.3 The Guideline for PaRO-DP Strategy Selection

While we propose a set of effective strategies in PaRO-DP, there remains a gap in applying a suitable strategy from our PaRO-DP across diverse training scenarios with different $\Psi$ and $\Psi'$. Therefore, we present a guideline based on quantitative calculations. We are dedicated to identifying the optimal partitioning states ($P_x$) that minimize the total time cost ($T$) of mini-batch training in a given scenario, as the throughput is inversely proportional to $T$. The total time cost $T$ contains the time of communication $t_{comm.}$ and the time of compute $t_{comp.}$, and $t_{overlap}$ denoting the overlapping time of communication and computation. It could be formalized as a optimization problem as illustrated in Formula 1.

$$\min_{P_{p+g+os}} t_{comm.} + t_{comp.} - t_{overlap} \Rightarrow \min_{P_{p+g+os}} \max \{t_{comm.}, t_{comp.}\} \tag{1}$$

Assuming that communication and computation can be fully overlapped, $T$ can be approximately regarded as the maximum of $t_{comm.}$ and $t_{comp.}$. Given a specific batch size, $t_{comp.}$ can be straightforwardly determined through a single forward computation. But the time of communication is more complex as shown in Formula 2.

$$t_{comm.} = (t_{param} + t_{gradient}) * s + t_{update} \tag{2}$$

For each micro-batch step, there is a time cost $t_{param}$ of all-gather for parameters in both the forward and backward passes and a time cost $t_{gradient}$ of reduce-scatter for gradients in the backward pass. For the last micro-batch of a mini-batch, there is a time cost, $t_{update}$, for reduce-scatter/all-reduce of gradients and all-gather for the updated parameters.

Therefore, the guideline implies that finding the partitioning states ($P_{p+g+os}$) in which minimizing $T$, as defined by Formula 1, is the recommended strategy for the given scenario. The process of calculation and selection is as follows, with further details in Appendix A.3:

1. Input: $n$, $m$, $B$, $B'$, $\Psi$, $\Psi'$, $s$ and $t_{comp.}$
2. Calculate the T values for various strategies.
3. Select the strategy with the smallest T value.

### 3.2 PaRO-CC

Distributed training on multiple groups of GPUs using existing methods often requires global collective communication operations. For example, ZeRO-3 uses a global all-gather to obtain parameters of the current layer and a global reduce-scatter to synchronize gradients. Megatron-TP uses a global all-reduce to synchronize the computation results of a split layer. Although PaRO-DP decreases the frequency of global communication operations by redesigning the training process, some operations remain necessary. For example, $P_{IGG}$ needs a global reduce-scatter to synchronize the gradients obtained from the backward pass of each layer.

In the Ring topology, a collective communication divides a collective communication into $n-1$ steps. In each step, each GPU is responsible for sending the currently relevant data block to the next GPU in

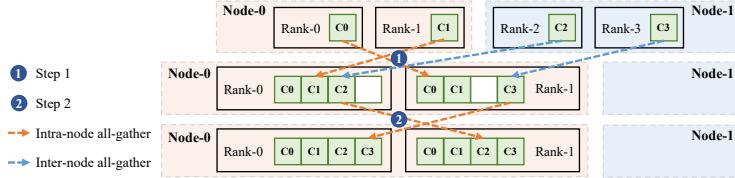

Figure 2: The communication stages of PaRO-CC. C* denotes a data chunk. Only the procedure of all-gather operation on Node 0 is shown for brevity. In step 1, inter- and intra-node all-gather are executed in parallel. In step 2, only inter-node all-gather is executed.

the ring. Clearly, in this process, each step includes both intra-group and inter-group communication, leading to the speed bottleneck of each step being on the inter-group communication.

Therefore, we introduce PaRO-CC to accelerate global collective communication operations by organizing GPUs into groups based on the network topology and rearranging the communication topology accordingly. Groups are configured to form an outer ring, with each group further organizing an inner ring. The operation of PaRO-CC is segmented into intra-group and inter-group parts, both of which can utilize the Ring topology. Consequently, within a single step, communication is confined to either intra-group or inter-group interactions, avoiding the need to address both simultaneously. It is straightforward to ascertain that for the all-gather operation, execution of the inter-group part should precede that of the intra-group part. Conversely, in the case of the reduce-scatter operation, the intra-group part should be executed before the inter-group part. Additionally, during the all-gather operation, the two parts of communication can be executed concurrently, assuming that suitable blocking points are established to manage the flow of data. Figure 2 illustrates an example of performing a global all-gather operation using PaRO-CC. More details about PaRO-CC can be found in Appendix A.2.

### 3.3 Applications of PaRO

PaRO can be employed independently as a basic strategy, or be used together with others in a composite strategy. For example, PaRO-DP can be an alternative method to data parallelism in hybrid parallelism, e.g. 4D parallelism, when training large LLMs in heterogeneous networks [29]. It can be orthogonally integrated with Sequence Parallelism [11, 14, 20]. At the same time, PaRO-CC can be applied in various distributed training strategies that require global collective communication operations, to accelerate the training. Furthermore, we argue that the comprehensive PaRO-DP strategy set provides more flexibility to complicated machine learning systems, such as distributed RLHF systems [24, 33, 37], where each sub-model has different memory or communication requirements.

## 4 Experiments and Analysis

### 4.1 Experiment Environments

Our experimental cluster consists of up to 16 DGX nodes as 16 groups, each containing 8 Ampere A100 SXM4 80GB GPUs. The GPUs in each node are interconnected via NVLink/NVSwitch with a bidirectional bandwidth of up to 2400Gbps. These nodes are connected through RDMA over Converged Ethernet (RoCE), which can provide 100Gbps of inter-node bandwidth. The software environment includes CUDA-11.7, DeepSpeed-v0.10.0, PyTorch-v1.9.2, Megatron-LM-v2.6 and NCCL-v2.14.3.

### 4.2 Experiment Settings

To evaluate the performance of PaRO-DP, we compared them with current SOTA methods, including ZeRO, MiCS, ZeRO++, and FSDP-hsz, across different model sizes (denoted as $\Psi$, such as 7B and 65B) and the number of trainable parameters ($\Psi'$, categorized as full-parameters, partial-parameters, and PEFT). For each scenario, we select the recommended PaRO-DP strategies based on the throughput indicator or TPS indicator $log(1/T)$, as outlined by the guideline in Section 3.1.3. The strategies of $P_{NN*}$ (ZeRO-1 as $P_{NNG}$, $P_{NNI}$ and DDP as $P_{NNN}$) were not considered due to its inability

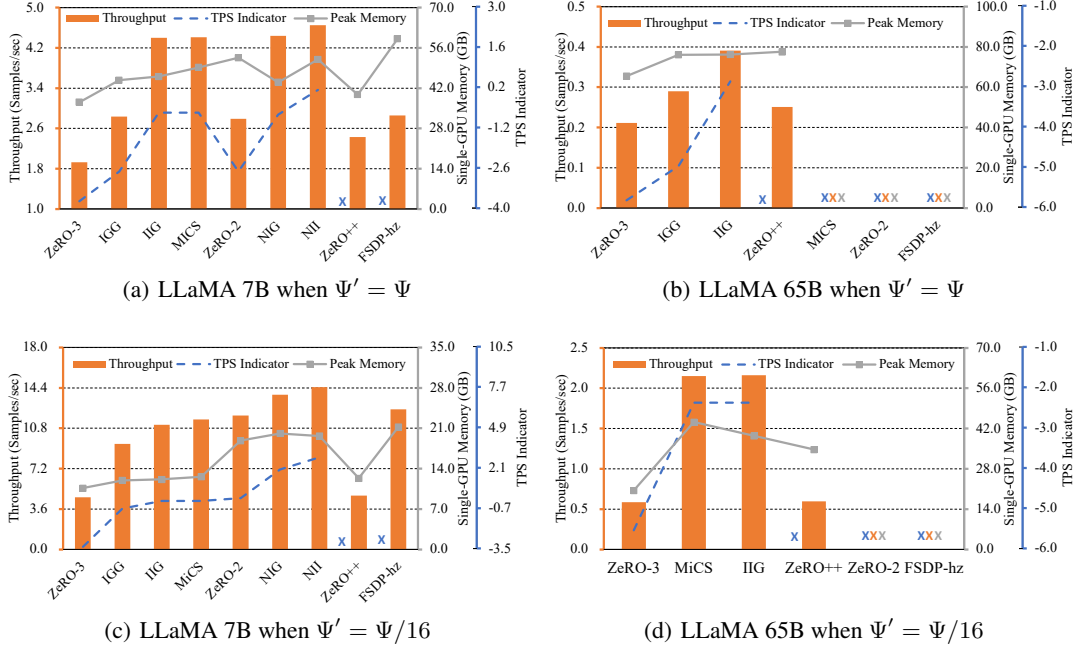

(a) LLaMA 7B when $\Psi' = \Psi$

(b) LLaMA 65B when $\Psi' = \Psi$

(c) LLaMA 7B when $\Psi' = \Psi/16$

(d) LLaMA 65B when $\Psi' = \Psi/16$

Figure 3: The throughput and memory usage during LLaMA training with varying trainable parameters ($\Psi'$). The blue dashed line represents the trend of the throughput indicator, represented by the TPS Indicator using $log(1/T)$, calculated based on the guideline. The cross indicates OOM.

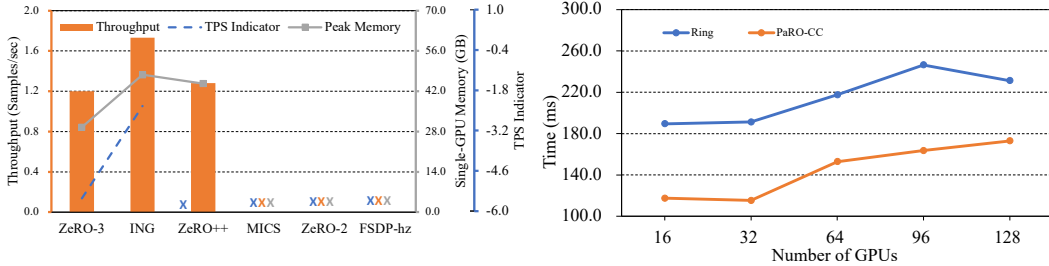

Figure 4: Throughput and Memory Usage of training LLaMA-65B in the PEFT($\Psi' = 3\Psi/1000$) scenario.

Figure 5: Collective Communication Time (millisecond/ms) with the increasing number of GPUs.

to run the large-scale model in our experiments. We use LLaMA-7B and LLaMA-65B [29] to evaluate the throughput and acceleration performance across 32 GPUs within 4 DGX nodes. For the LLaMA-65B model, we activate checkpointing to ensure successful training. The C4 corpus in RedPajama[5] was used as the training data. During training, we set the sequence length to 512, and the effective total batch size to 1280 (with 10 as gradient accumulation steps) in a mixed precision training manner.

## 4.3 Efficiency of PaRO-DP

Incorporating group and refined partitioning, the PaRO-DP strategies adeptly distribute the volume of inter- and intra-group communication to minimize overall communication costs. This results in a significant boost by up to 266% in training speed, accomplished with acceptable memory increment. The following section will describe the experimental results from various scenarios.

**Full-parameters training:** In this scenario ($\Psi' = \Psi$), Figure 3(a) shows that the guideline recommends strategies such as $P_{NII}$, $P_{IIG}$ and $P_{III}$ (MiCS), which perform better throughput than others

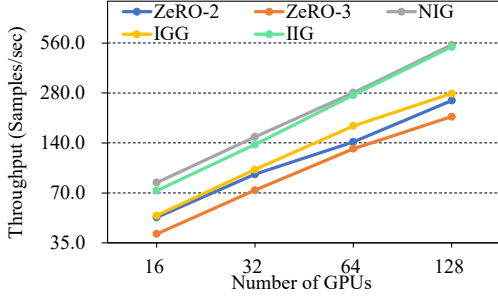
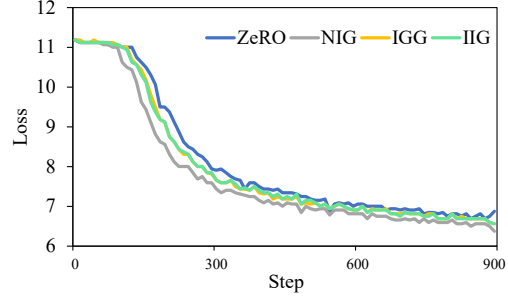

Figure 6: The throughput (samples/sec) when increasing the scale of GPUs.

Figure 7: Training convergence of PaRO against ZeRO using LLaMA-7B.

in LLaMA-7B. The throughput of $P_{IIG}$ is improved by 128% than ZeRO-3, and the throughput of $P_{NII}$ is improved by 67% than ZeRO-2. While in LLaMA-65B, Figure 3(b) shows the throughput of $P_{IIG}$ is improved by 86% than ZeRO-3. Since LLaMA-65B requires more fine-grained partitioning, only ZeRO-3, ZeRO++, $P_{IGG}$ and $P_{IIG}$ can perform training, while other solutions suffer from out-of-memory (OOM) issues.

**Partial-parameters training:** In experiments of partial-parameters training ($\Psi' = \Psi/16$), we also reached a consistent conclusion. As shown in Figure 3(c), in LLaMA-7B, the throughput of $P_{NIG}$ and $P_{NII}$ is increased by 15% and 21% compared to ZeRO-2. Figure 3(d) shows the throughput of $P_{III}$ (MiCS) and $P_{IIG}$ is improved by 264% and 266% compared with ZeRO-3 in LLaMA-65B, and $P_{IIG}$ uses less memory than MiCS.

**PEFT:** In PEFT scenarios ($\Psi' = 3\Psi/1000$) with LLaMA-65B shown as Figure 4, the throughput of $P_{ING}$ is improved by 44% compared to ZeRO-3.

**Accuracy of the Guideline:** The selection of high-throughput strategies across different scenarios is determined by calculations based on the guideline. It chooses the strategy that owns a higher throughput indicator, $log(1/T)$, compared to the baseline. The correlation between the guideline and actual performance in these scenarios is apparent, confirming that the throughput indicator reliably reflects the real world.

## 4.4 Efficiency of PaRO-CC

In the section, we performed experiments using up to 16 DGX nodes with a communication volume of 1GB. We measured the communication time of the all-gather operation with the traditional Ring (baseline) and PaRO-CC, shown in Figure 5. Compared with the traditional Ring, there is a pronounced performance enhancement by 25.2%-39.7% with different cluster scales (from 16 to 128 GPUs). An additional experiment with 4 DGX nodes reveals that incorporating PaRO-CC optimization into end-to-end training tasks using Megatron-TP results in a 17% increase in throughput, from 129 to 152 samples/second. Therefore, PaRO-CC can significantly improve communication efficiency by rearranging the communication topology.

## 4.5 Near-linear Scalability

To investigate the scalability of PaRO, we analyze the throughput of PaRO and ZeRO under varying counts of GPUs, as depicted in Figure 6. The experiments utilized the LLaMA-7B model. Using identical GPU resource configurations, PaRO consistently exceeded the speedup ratio of both ZeRO-2 and ZeRO-3 baselines. Notably, the speedup ratio for $P_{IIG}$ and $P_{NIG}$ from 16 to 128 GPUs are 0.92 and 0.84 respectively. These values are closer to 1, indicating near-linear scalability, compared to the speedup ratio of 0.63 for both ZeRO-2 and ZeRO-3.

## 4.6 Model Convergence

In this section, we demonstrate the consistent convergence of PaRO compared to ZeRO, which validates the correctness of our system. We used LLaMA-7B and C4 corpus in RedPajama to evaluate

the convergence of PaRO with 16 GPUs. During training, we set the sequence length to 128, the effective batch size to 1024, and the number of gradient accumulation steps to 8. The loss validation process aims not to produce the same loss as ZeRO but to ensure identical convergence behaviours. As shown in Figure 7, PaRO provides the same convergence as ZeRO. The vertical axis represents the training step, but the duration per step varies with different strategies, affecting the training throughput.

## 5  Conclusion

In this paper, we dive into the trade-off between memory consumption and communication costs across various training scenarios in data parallelism training. Considering the performance gap between intra- and inter-group networks, we introduce PaRO, a refined and flexible partition strategy set. It contains the basic strategy, PaRO-DP, along with an effective collective communication topology, PaRO-CC. In heterogeneous networks, PaRO's advantages are more prominently highlighted due to its effective adaption to the limited inter-group network, which outperforms the ZeRO-3 by up to 266%. Furthermore, we conducted a comprehensive quantitative analysis of this trade-off and established a guideline to assist in selecting the suitable distributed training strategy in different scenarios.

## 6  Limitations and Societal Impacts

**Limitations.** Our method primarily focuses on data parallelism in training large language models. To train extremely large models, it is recommended to integrate our PaRO strategy with other parallelism techniques, such as tensor and pipeline parallelism, though this may reduce the ease of use. Furthermore, performance improvements of our method are largely due to the efficient utilization of heterogeneous networks, limiting its applicability in homogeneous networks.

**Societal Impacts.** By detailing our methodology and releasing our code, we aim to advance both research and industrial practices in the field of large language model training. By improving training efficiency, PaRO has the potential to reduce the environmental impact of training LLMs, which often require dozens to thousands of GPU devices. Additionally, our PaRO strategy is the first optimized distributed strategy for fine-tuning scenarios, especially PEFT scenarios. This enhances our social impact by benefiting large companies and democratizing the training of LLMs using a limited number of GPUs.

## Acknowledgements

We thank the anonymous reviewers for their useful comments and suggestions, and we also thank Shangchun Zhao for his insightful advice.

# References

[1] Gibiansky Andrew. Bringing hpc techniques to deep learning, 2017. URL `https://andrew.gibiansky.com/blog/machine-learning/baidu-allreduce`.

[2] Zhengda Bian, Qifan Xu, Boxiang Wang, and Yang You. Maximizing parallelism in distributed training for huge neural networks. *CoRR*, abs/2105.14450, 2021. URL `https://arxiv.org/abs/2105.14450`.

[3] Tom B. Brown, Benjamin Mann, Nick Ryder, Melanie Subbiah, Jared Kaplan, et al. Language models are few-shot learners. In Hugo Larochelle, Marc'Aurelio Ranzato, et al., editors, *Advances in Neural Information Processing Systems 33: Annual Conference on Neural Information Processing Systems 2020, NeurIPS 2020, December 6-12, 2020, virtual*, 2020. URL `https://proceedings.neurips.cc/paper/2020/hash/1457c0d6bfcb4967418bfb8ac142f64a-Abstract.html`.

[4] Aakanksha Chowdhery, Sharan Narang, Jacob Devlin, Maarten Bosma, Gaurav Mishra, et al. Palm: Scaling language modeling with pathways. *J. Mach. Learn. Res.*, 24:240:1–240:113, 2023. URL `http://jmlr.org/papers/v24/22-1144.html`.

[5] Together Computer. Redpajama: an open dataset for training large language models, October 2023. URL `https://github.com/togethercomputer/RedPajama-Data`.

[6] Tri Dao, Daniel Y. Fu, Stefano Ermon, Atri Rudra, and Christopher Ré. Flashattention: Fast and memory-efficient exact attention with io-awareness. In Sanmi Koyejo, S. Mohamed, et al., editors, *Advances in Neural Information Processing Systems 35: Annual Conference on Neural Information Processing Systems 2022, NeurIPS 2022, New Orleans, LA, USA, November 28 - December 9, 2022*, 2022. URL `http://papers.nips.cc/paper_files/paper/2022/hash/67d57c32e20fd0a7a302cb81d36e40d5-Abstract-Conference.html`.

[7] Tim Dettmers, Artidoro Pagnoni, Ari Holtzman, and Luke Zettlemoyer. Qlora: Efficient finetuning of quantized llms. In Alice Oh, Tristan Naumann, et al., editors, *Advances in Neural Information Processing Systems 36: Annual Conference on Neural Information Processing Systems 2023, NeurIPS 2023, New Orleans, LA, USA, December 10 - 16, 2023*, 2023. URL `http://papers.nips.cc/paper_files/paper/2023/hash/1feb87871436031bdc0f2beaa62a049b-Abstract-Conference.html`.

[8] Zhengxiao Du, Yujie Qian, Xiao Liu, Ming Ding, Jiezhong Qiu, et al. GLM: general language model pretraining with autoregressive blank infilling. In Smaranda Muresan, Preslav Nakov, and Aline Villavicencio, editors, *Proceedings of the 60th Annual Meeting of the Association for Computational Linguistics (Volume 1: Long Papers), ACL 2022, Dublin, Ireland, May 22-27, 2022*, pages 320–335. Association for Computational Linguistics, 2022. doi: 10.18653/V1/2022.ACL-LONG.26. URL `https://doi.org/10.18653/v1/2022.acl-long.26`.

[9] Abhimanyu Dubey, Abhinav Jauhri, Abhinav Pandey, Abhishek Kadian, Ahmad Al-Dahle, et al. The llama 3 herd of models. *arXiv preprint arXiv:2407.21783*, 2024.

[10] Shiqing Fan, Yi Rong, Chen Meng, Zongyan Cao, Siyu Wang, et al. DAPPLE: a pipelined data parallel approach for training large models. In Jaejin Lee and Erez Petrank, editors, *PPoPP '21: 26th ACM SIGPLAN Symposium on Principles and Practice of Parallel Programming, Virtual Event, Republic of Korea, February 27- March 3, 2021*, pages 431–445. ACM, 2021. doi: 10.1145/3437801.3441593. URL `https://doi.org/10.1145/3437801.3441593`.

[11] Diandian Gu, Peng Sun, Qinghao Hu, Ting Huang, Xun Chen, Yingtong Xiong, Guoteng Wang, Qiaoling Chen, Shangchun Zhao, Jiarui Fang, et al. Loongtrain: Efficient training of long-sequence llms with head-context parallelism. *arXiv preprint arXiv:2406.18485*, 2024.

[12] Edward J. Hu, Yelong Shen, Phillip Wallis, Zeyuan Allen-Zhu, Yuanzhi Li, et al. Lora: Low-rank adaptation of large language models. In *The Tenth International Conference on Learning Representations, ICLR 2022, Virtual Event, April 25-29, 2022*. OpenReview.net, 2022. URL `https://openreview.net/forum?id=nZeVKeeFYf9`.

[13] Yanping Huang, Youlong Cheng, Ankur Bapna, Orhan Firat, Dehao Chen, et al. Gpipe: Efficient training of giant neural networks using pipeline parallelism, 2019. URL `https://proceedings.neurips.cc/paper/2019/hash/093f65e080a295f8076b1c5722a46aa2-Abstract.html`.

[14] Sam Ade Jacobs, Masahiro Tanaka, Chengming Zhang, Minjia Zhang, Shuaiwen Leon Song, et al. Deepspeed ulysses: System optimizations for enabling training of extreme long sequence transformer models. *arXiv preprint arXiv:2309.14509*, 2023.

[15] Zhihao Jia, Matei Zaharia, and Alex Aiken. Beyond data and model parallelism for deep neural networks. In Ameet Talwalkar, Virginia Smith, and Matei Zaharia, editors, *Proceedings of the SysML Conference 2019 (SysML 2019), Stanford, CA, USA, March 31 - April 2, 2019*. mlsys.org, 2019. URL `https://proceedings.mlsys.org/paper_files/paper/2019/hash/b422680f3db0986ddd7f8f126baaf0fa-Abstract.html`.

[16] Diederik P. Kingma and Jimmy Ba. Adam: A method for stochastic optimization. In Yoshua Bengio and Yann LeCun, editors, *3rd International Conference on Learning Representations, ICLR 2015, San Diego, CA, USA, May 7-9, 2015, Conference Track Proceedings*, 2015. URL `http://arxiv.org/abs/1412.6980`.

[17] Dmitry Lepikhin, HyoukJoong Lee, Yuanzhong Xu, Dehao Chen, Orhan Firat, et al. Gshard: Scaling giant models with conditional computation and automatic sharding. In *9th International Conference on Learning Representations, ICLR 2021, Virtual Event, Austria, May 3-7, 2021*. OpenReview.net, 2021. URL `https://openreview.net/forum?id=qrwe7XHTmYb`.

[18] Shen Li, Yanli Zhao, Rohan Varma, Omkar Salpekar, Pieter Noordhuis, et al. Pytorch distributed: Experiences on accelerating data parallel training. *Proc. VLDB Endow.*, 13(12):3005–3018, 2020. doi: 10.14778/3415478.3415530. URL `http://www.vldb.org/pvldb/vol13/p3005-li.pdf`.

[19] Shenggui Li, Hongxin Liu, Zhengda Bian, Jiarui Fang, Haichen Huang, et al. Colossal-ai: A unified deep learning system for large-scale parallel training. In *Proceedings of the 52nd International Conference on Parallel Processing, ICPP 2023, Salt Lake City, UT, USA, August 7-10, 2023*, pages 766–775. ACM, 2023. doi: 10.1145/3605573.3605613. URL `https://doi.org/10.1145/3605573.3605613`.

[20] Hao Liu, Matei Zaharia, and Pieter Abbeel. Ring attention with blockwise transformers for near-infinite context. *arXiv preprint arXiv:2310.01889*, 2023.

[21] Haokun Liu, Derek Tam, Mohammed Muqeeth, Jay Mohta, Tenghao Huang, et al. Few-shot parameter-efficient fine-tuning is better and cheaper than in-context learning. In Sanmi Koyejo, S. Mohamed, et al., editors, *Advances in Neural Information Processing Systems 35: Annual Conference on Neural Information Processing Systems 2022, NeurIPS 2022, New Orleans, LA, USA, November 28 - December 9, 2022*, 2022. URL `http://papers.nips.cc/paper_files/paper/2022/hash/0cde695b83bd186c1fd456302888454c-Abstract-Conference.html`.

[22] Xupeng Miao, Yujie Wang, Youhe Jiang, Chunan Shi, Xiaonan Nie, et al. Galvatron: Efficient transformer training over multiple gpus using automatic parallelism. *Proc. VLDB Endow.*, 16(3):470–479, 2022. doi: 10.14778/3570690.3570697. URL `https://www.vldb.org/pvldb/vol16/p470-miao.pdf`.

[23] Deepak Narayanan, Mohammad Shoeybi, Jared Casper, Patrick LeGresley, Mostofa Patwary, et al. Efficient large-scale language model training on GPU clusters using megatron-lm. In Bronis R. de Supinski, Mary W. Hall, and Todd Gamblin, editors, *International Conference for High Performance Computing, Networking, Storage and Analysis, SC 2021, St. Louis, Missouri, USA, November 14-19, 2021*, page 58. ACM, 2021. doi: 10.1145/3458817.3476209. URL `https://doi.org/10.1145/3458817.3476209`.

[24] Long Ouyang, Jeffrey Wu, Xu Jiang, Diogo Almeida, Carroll L. Wainwright, et al. Training language models to follow instructions with human feedback. In Sanmi Koyejo, S. Mohamed, et al., editors, *Advances in Neural Information Processing Systems 35: Annual Conference on Neural Information Processing Systems 2022, NeurIPS 2022, New Orleans, LA, USA, November 28 - December 9, 2022*, 2022. URL `http://papers.nips.cc/paper_files/paper/2022/hash/b1efde53be364a73914f58805a001731-Abstract-Conference.html`.

[25] Samyam Rajbhandari, Jeff Rasley, Olatunji Ruwase, and Yuxiong He. Zero: memory optimizations toward training trillion parameter models. In Christine Cuicchi, Irene Qualters, and William T. Kramer, editors, *Proceedings of the International Conference for High Performance Computing, Networking, Storage and Analysis, SC 2020, Virtual Event / Atlanta, Georgia, USA, November 9-19, 2020*, page 20. IEEE/ACM, 2020. doi: 10.1109/SC41405.2020.00024. URL `https://doi.org/10.1109/SC41405.2020.00024`.

[26] Noam Shazeer, Youlong Cheng, Niki Parmar, Dustin Tran, Ashish Vaswani, et al. Mesh-tensorflow: Deep learning for supercomputers. In Samy Bengio, Hanna M. Wallach, et al., editors, *Advances in Neural Information Processing Systems 31: Annual Conference on Neural Information Processing Systems 2018, NeurIPS 2018, December 3-8, 2018, Montréal, Canada*, pages 10435–10444, 2018. URL `https://proceedings.neurips.cc/paper/2018/hash/3a37abdeefe1dab1b30f7c5c7e581b93-Abstract.html`.

[27] Mohammad Shoeybi, Mostofa Patwary, Raul Puri, Patrick LeGresley, Jared Casper, and Bryan Catanzaro. Megatron-lm: Training multi-billion parameter language models using model parallelism. *CoRR*, abs/1909.08053, 2019. URL `http://arxiv.org/abs/1909.08053`.

[28] Shaden Smith, Mostofa Patwary, Brandon Norick, Patrick LeGresley, Samyam Rajbhandari, et al. Using deepspeed and megatron to train megatron-turing NLG 530b, A large-scale generative language model. *CoRR*, abs/2201.11990, 2022. URL `https://arxiv.org/abs/2201.11990`.

[29] Hugo Touvron, Thibaut Lavril, Gautier Izacard, Xavier Martinet, Marie-Anne Lachaux, et al. Llama: Open and efficient foundation language models. *CoRR*, abs/2302.13971, 2023. doi: 10.48550/ARXIV.2302. 13971. URL `https://doi.org/10.48550/arXiv.2302.13971`.

[30] Colin Unger, Zhihao Jia, Wei Wu, Sina Lin, Mandeep Baines, et al. Unity: Accelerating DNN training through joint optimization of algebraic transformations and parallelization. In Marcos K. Aguilera and Hakim Weatherspoon, editors, *16th USENIX Symposium on Operating Systems Design and Implementation, OSDI 2022, Carlsbad, CA, USA, July 11-13, 2022*, pages 267–284. USENIX Association, 2022. URL `https://www.usenix.org/conference/osdi22/presentation/unger`.

[31] Boxiang Wang, Qifan Xu, Zhengda Bian, and Yang You. Tesseract: Parallelize the tensor parallelism efficiently. In *Proceedings of the 51st International Conference on Parallel Processing, ICPP 2022, Bordeaux, France, 29 August 2022 - 1 September 2022*, pages 12:1–12:11. ACM, 2022. doi: 10.1145/ 3545008.3545087. URL `https://doi.org/10.1145/3545008.3545087`.

[32] Guanhua Wang, Heyang Qin, Sam Ade Jacobs, Connor Holmes, Samyam Rajbhandari, et al. Zero++: Extremely efficient collective communication for giant model training, 2023. URL `https://doi.org/10.48550/arXiv.2306.10209`.

[33] Youshao Xiao, Weichang Wu, Zhenglei Zhou, Fagui Mao, Shangchun Zhao, et al. An adaptive placement and parallelism framework for accelerating RLHF training. *CoRR*, abs/2312.11819, 2023. doi: 10.48550/ ARXIV.2312.11819. URL `https://doi.org/10.48550/arXiv.2312.11819`.

[34] Youshao Xiao, Shangchun Zhao, Zhenglei Zhou, Zhaoxin Huan, Lin Ju, et al. G-meta: Distributed meta learning in GPU clusters for large-scale recommender systems. In Ingo Frommholz, Frank Hopfgartner, et al., editors, *Proceedings of the 32nd ACM International Conference on Information and Knowledge Management, CIKM 2023, Birmingham, United Kingdom, October 21-25, 2023*, pages 4365–4369. ACM, 2023. doi: 10.1145/3583780.3615208. URL `https://doi.org/10.1145/3583780.3615208`.

[35] Qifan Xu and Yang You. An efficient 2d method for training super-large deep learning models. In *IEEE International Parallel and Distributed Processing Symposium, IPDPS 2023, St. Petersburg, FL, USA, May 15-19, 2023*, pages 222–232. IEEE, 2023. doi: 10.1109/IPDPS54959.2023.00031. URL `https://doi.org/10.1109/IPDPS54959.2023.00031`.

[36] Yuanzhong Xu, HyoukJoong Lee, Dehao Chen, Blake A. Hechtman, Yanping Huang, et al. GSPMD: general and scalable parallelization for ML computation graphs. *CoRR*, abs/2105.04663, 2021. URL `https://arxiv.org/abs/2105.04663`.

[37] Zhewei Yao, Reza Yazdani Aminabadi, Olatunji Ruwase, Samyam Rajbhandari, Xiaoxia Wu, et al. Deepspeed-chat: Easy, fast and affordable RLHF training of chatgpt-like models at all scales. *CoRR*, abs/2308.01320, 2023. doi: 10.48550/ARXIV.2308.01320. URL `https://doi.org/10.48550/arXiv.2308.01320`.

[38] Longteng Zhang, Lin Zhang, Shaohuai Shi, Xiaowen Chu, and Bo Li. Lora-fa: Memory-efficient low-rank adaptation for large language models fine-tuning. *CoRR*, abs/2308.03303, 2023. doi: 10.48550/ARXIV. 2308.03303. URL `https://doi.org/10.48550/arXiv.2308.03303`.

[39] Zhen Zhang, Shuai Zheng, Yida Wang, Justin Chiu, George Karypis, et al. Mics: Near-linear scaling for training gigantic model on public cloud. *Proc. VLDB Endow.*, 16(1):37–50, 2022. doi: 10.14778/3561261. 3561265. URL `https://www.vldb.org/pvldb/vol16/p37-zhang.pdf`.

[40] Yanli Zhao, Andrew Gu, Rohan Varma, Liang Luo, Chien-Chin Huang, et al. Pytorch FSDP: experiences on scaling fully sharded data parallel. *Proc. VLDB Endow.*, 16(12):3848–3860, 2023. doi: 10.14778/ 3611540.3611569. URL `https://www.vldb.org/pvldb/vol16/p3848-huang.pdf`.

[41] Lianmin Zheng, Zhuohan Li, Hao Zhang, Yonghao Zhuang, Zhifeng Chen, et al. Alpa: Automating inter- and intra-operator parallelism for distributed deep learning. In Marcos K. Aguilera and Hakim Weatherspoon, editors, *16th USENIX Symposium on Operating Systems Design and Implementation, OSDI 2022, Carlsbad, CA, USA, July 11-13, 2022*, pages 559–578. USENIX Association, 2022. URL `https://www.usenix.org/conference/osdi22/presentation/zheng-lianmin`.

# A Appendix

## A.1 Implementation of PaRO-DP

This section provides an overview of the implementation strategies within the PaRO-DP. It details the partitioning of model states and elucidates the collective communications used during data-parallel training. The following subsections evaluate partitioning effectiveness and proceed with in-depth descriptions and pseudo-codes for each PaRO-DP strategy.

### A.1.1 Effective Strategies of Partitioning

Table 1 presents all the combinations of model states partitioning $P_{p+g+os}$, with ticks indicating those combinations that constitute effective solutions. The cross mark indicates the 13 strategies that have been eliminated by the insight in Section 3.1.1.

Table 1: The 27 combinations of model states partitioning.

| Strategy of Partition ($P_{p+g+os}$) | Effective | Strategy of Partition ($P_{p+g+os}$) | Effective |
|---|---|---|---|
| NNN(DDP) | ✓ | IIG | ✓ |
| NNI | ✓ | IGN | ✗ |
| NNG(ZeRO-1) | ✓ | IGI | ✗ |
| NIN | ✗ | IGG | ✓ |
| NII | ✓ | GNN | ✗ |
| NIG | ✓ | GNI | ✗ |
| NGN | ✗ | GNG | ✓ |
| NGI | ✗ | GIN | ✗ |
| NGG(ZeRO-2) | ✓ | GII | ✗ |
| INN | ✗ | GIG | ✓ |
| INI | ✓ | GGN | ✗ |
| ING | ✓ | GGI | ✗ |
| IIN | ✗ | GGG(ZeRO-3) | ✓ |
| III(MiCS) | ✓ | | |

### A.1.2 Collective Communication Used in PaRO-DP

We divide the data evenly into $n$ (the number of all GPUs) blocks, and the blocks held by GPU with different partitioning are shown as Table 2. There are some additional notations:

$i$: The index of the current group in all groups, $0 \leq i < ng$

$j$: The index of the current GPU in the group, $0 \leq j < m$

$GPU_{i,j}$: The j-th GPU in the i-th group

$blocks$: The array with the length is $n$ of all blocks

Table 2: The partitioned blocks of data held by GPU

| Partitioning Strategy | Data blocks held by $GPU_{i,j}$ |
|---|---|
| No partitioning ($N$) | $blocks[0:n]$ |
| Intra-group partitioning ($I$) | $blocks[j \times ng : (j+1) \times ng]$ |
| Global partitioning ($G$) | $blocks[j \times ng + i]$ |

The following Table 3 introduces the collective communication operations used for synchronization between blocks of different partitioning.

Table 3: The Collective Communication used in PaRO-DP from the perspective of a single GPU($GPU_{i_0,j_0}$). The values ($N/I/G$) of Inputs/Outputs blocks refer to Tabel 2.

| Collective Communication | Input blocks | Output blocks | Participation ranks ($GPU_{i,j}$) | Description |
|---|---|---|---|---|
| global_all_gather | $G$ | $N$ | $\{0 \le i < g, 0 \le j < m\}$ | optimized by PaRO-CC |
| global_reduce_scatter | $N$ | $G$ | $\{0 \le i < g, 0 \le j < m\}$ | optimized by PaRO-CC |
| global_all_reduce | $N$ | $N$ | $\{0 \le i < g, 0 \le j < m\}$ | - |
| intra_group_all_gather | $I$ | $N$ | $\{i = i_0, 0 \le j < m\}$ | - |
| intra_group_reduce_scatter | $N$ | $I$ | $\{i = i_0, 0 \le j < m\}$ | - |
| inter_group_all_gather | $G$ | $I$ | $\{0 \le i < g, j = j_0\}$ | all-gather with $j_0$-th GPUs from different groups. |
| inter_group_reduce_scatter | $I$ | $G$ | $\{0 \le i < g, j = j_0\}$ | reduce-scatter with $j_0$-th GPUs from different groups. |
| inter_group_all_reduce | $I$ | $I$ | $\{0 \le i < g, j = j_0\}$ | all-reduce with $j_0$-th GPUs from different groups. |

### A.1.3 Pseudo-code of PaRO-DP

We illustrate the PaRO-DP algorithms in this section. Each algorithm corresponds to a specific PaRO-DP strategy.

---

**Algorithm 1:** $P_{IIG}$ Algorithm

---
**Input:** $model, data$
**Output:** $model$

1 **for** *mini-batch in epoch* **do**
2      **for** *micro-batch in mini-batch* **do**
3          **for** *layer in model.layers()* **do**
4              **intra_group_all_gather**(layer.parameters);
5              layer.forward();
6          **for** *layer in model.reverse_layers()* **do**
7              **intra_group_all_gather**(layer.parameters);
8              layer.backward();
9              **intra_group_reduce_scatter**(layer.gradients);
10              layer.gradients.accumulate();
11      **inter_group_reduce_scatter**(model.gradients);
12      optimizer.step(); // Apply gradient
13      **inter_group_all_gather**(model.parameters);

---

**Algorithm 2:** $P_{IGG}$ Algorithm

**Input:** *model, data*
**Output:** *model*

1  **for** *mini-batch in epoch* **do**
2      **for** *micro-batch in mini-batch* **do**
3          **for** *layer in model.layers()* **do**
4              **intra_group_all_gather**(layer.parameters);
5              layer.forward();
6          **for** *layer in model.reverse_layers()* **do**
7              **intra_group_all_gather**(layer.parameters);
8              layer.backward();
9              **global_reduce_scatter**(layer.gradients);
10             layer.gradients.accumulate();
11     optimizer.step(); `// Apply gradient`
12     **inter_group_all_gather**(model.parameters);

---

**Algorithm 3:** $P_{NIG}$ Algorithm

**Input:** *model, data*
**Output:** *model*

1  **for** *mini-batch in epoch* **do**
2      **for** *micro-batch in mini-batch* **do**
3          **for** *layer in model.layers()* **do**
4              layer.forward();
5          **for** *layer in model.reverse_layers()* **do**
6              layer.backward();
7              **intra_group_reduce_scatter**(layer.gradients);
8              layer.gradients.accumulate();
9      **inter_group_reduce_scatter**(model.gradients);
10     optimizer.step(); `// Apply gradient`
11     **global_all_gather**(model.parameters);

---

**Algorithm 4:** $P_{NII}$ Algorithm

**Input:** *model, data*
**Output:** *model*

1  **for** *mini-batch in epoch* **do**
2      **for** *micro-batch in mini-batch* **do**
3          **for** *layer in model.layers()* **do**
4              layer.forward();
5          **for** *layer in model.reverse_layers()* **do**
6              layer.backward();
7              **intra_group_reduce_scatter**(layer.gradients);
8              layer.gradients.accumulate();
9      **inter_group_all_reduce**(model.gradients);
10     optimizer.step(); `// Apply gradient`
11     **intra_group_all_gather**(model.parameters);

---

**Algorithm 5:** $P_{ING}$ Algorithm

---
**Input:** $model, data$
**Output:** $model$

1 **for** *mini-batch in epoch* **do**
2    **for** *micro-batch in mini-batch* **do**
3       **for** *layer in model.layers()* **do**
4          **intra_group_all_gather**(layer.parameters);
5          layer.forward();

6       **for** *layer in model.reverse_layers()* **do**
7          **intra_group_all_gather**(layer.parameters);
8          layer.backward();
9          layer.gradients.accumulate();

10    **global_reduce_scatter**(model.gradients);
11    optimizer.step(); // Apply gradient
12    **inter_group_all_gather**(model.parameters);

---

The implementations of other strategies of PaRO-DP can refer to the pseudo-code provided for the above-mentioned strategies, with the differences being in the details of how partitions are handled, which will not be reiterated here.

## A.2 Detail of PaRO-CC

Algorithm 6 and Algorithm 7 are pseudocodes for PaRO-CC All-Gather and Reduce-Scatter. Similar to traditional methods, in networks without NVIDIA SHARP configured, an all-reduce operation consists of one Reduce-Scatter operation followed by one All-Gather operation.

---

**Algorithm 6:** PaRO-CC All-Gather Algorithm

---
**Input:** Number of groups $ng$; Number of gpus in each group $m$; Current GPU index: $i, j$ ranges from $0, 0$ to $ng - 1, m - 1$

1 **Function** *inter_group_part***:**
2    **for** $s$=0 *to* $ng - 2$ **do**
3       **do in parallel**
4          send block[$(ng - s + i) * m\%ng + j$] to GPU[$(i + 1)\%ng, j$];
5          receive block[$(ng - s + i - 1) * m\%ng + j$] from GPU[$(i - 1 + ng)\%ng, j$];
6       mark block[$(ng - s + i - 1) * m\%ng + j$] as *ready*;

7 **Function** *intra_group_part***:**
8    **do**
9       select a *ready* data block or **wait**;
10       send the block to the next GPU in the inner ring;
11       receive the block from the previous CPU in the inner ring;
12       mark the received block as *ready*;
13    **while** *not finished*

14
15 split data into blocks as block[$0..ng * m - 1$];
16 mark block[$i * m + j$] as *ready*;
17 **do in parallel**
18    inter_group_part;
19    intra_group_part;

---

| Algorithm 7: PaRO-CC Reduce-Scatter Algorithm |
|---|
| **Input:** Number of groups $ng$; Number of gpus in each group $m$; Current GPU index: $i, j$ ranges from $0, 0$ to $ng - 1, m - 1$ |
| 1   perform reduce-scatter using inner ring; |
| 2   perform reduce-scatter using outer ring; |

## A.3   Detail of the Guideline

In this section, we present a detailed explanation of the time of communication cost $t_{comm.}$ calculation formula, employing symbols that align with the definitions provided in the main text. And give an example of the guideline based on the calculation formula.

### A.3.1   Detail of the Calculation Formula

$$t_{comm.} = (t_{param} + t_{gradient}) * s + t_{update}$$

Table 4 and Table 5 show the calculation formula of $t_{param}$ and $t_{gradient}$ for various partitioning states.

Table 4: The calculation formula of $t_{param}$

| $P_p$ | time | Description |
|---|---|---|
| $N$ | $0$ | |
| $I$ | $2 * \Psi/m * (m-1)/B'$ | intra-group all-gather parameters when forward and backward |
| $G$ | $2 * \Psi/n * (n-1)/B$ | global all-gather parameters when forward and backward |

Table 5: The calculation formula of $t_{gradient}$

| $P_g$ | time | Description |
|---|---|---|
| $N$ | $0$ | |
| $I$ | $\Psi'/m * (m-1)/B'$ | intra-group reduce-scatter gradients when backward |
| $G$ | $\Psi'/n * (n-1)/B$ | global reduce-scatter gradients when backward |

$$t_{update} = t_{sync_g} + t_{sync_p}$$

The $t_{update}$ includes two parts: the time of synchronizing gradients ($t_{sync_g}$, shown as Table 6) required for the optimizer states step and the time of synchronizing model parameters ($t_{sync_p}$, shown as Table 7) after updated.

Table 6: The calculation formula of $t_{sync_g}$

| $P_g$ | $P_{os}$ | time | Description |
|---|---|---|---|
| $N$ | $N$ | $2 * \Psi'/n * (n-1)/B$ | global all-reduce gradients |
| $N$ | $I$ | $\Psi'/n * (n-1)/B + \Psi'/m/g * (g-1)/B$ | inter-group reduce-scatter and all-reduce gradients |
| $N$ | $G$ | $\Psi'/n * (n-1)/B$ | global reduce-scatter gradients |
| $I$ | $I$ | $2 * \Psi'/m/g * (g-1)/B$ | inter-group all-reduce gradients |
| $I$ | $G$ | $\Psi'/m/g * (g-1)/B$ | inter-group reduce-scatter gradients |
| - | - | $0$ | ohters |

In light of our experimental configuration and environment, the salient parameters informing our derived guidelines include: $m = 32$, $n = 8$, $g = n/m = 4$, $s = 10$, $\Psi = 7 * 1e9$ or $65 * 1e9$, $B = 80 * 1e9$ (80Gbps), $B' = 2000 * 1e9$ (2000 Gbps), as determined through testing within our experimental GPU cluster.

Table 7: The calculation formula of $t_{sync_p}$

| $P_p$ | $P_{os}$ | time | Description |
|---|---|---|---|
| $N$ | $I$ | $\Psi'/m*(m-1)/B'$ | intra-group all-gather updated parameters |
| $N$ | $G$ | $\Psi'/n*(n-1)/B$ | global all-gather updated parameters |
| $I$ | $G$ | $\Psi'/m/g*(g-1)/B$ | inter-group all-gather updated parameters |
| - | - | $0$ | ohters |

Table 8: The Metric $(1/T)$ of training throughput and GPU memory of LLaMA-7B. $\Psi' = \Psi$ and $\Psi' = \Psi/16$ mean the different ratios of trainable parameters to model parameters.

| Strategy ($p/g/os$) | $\Psi' = \Psi$ | | $\Psi' = \Psi/16$ | |
|---|---|---|---|---|
| | $1/T$ | Mem(GB) | $1/T$ | Mem(GB) |
| NII(PaRO) | **1.151** | 24.447 | **17.827** | 13.752 |
| NIG(PaRO) | 0.489 | 17.113 | **7.716** | 13.293 |
| NGG(ZeRO-2) | 0.067 | 15.891 | 1.066 | 13.217 |
| INI | 0.386 | 24.447 | 0.841 | 3.056 |
| ING(PaRO) | 0.386 | 17.113 | 0.841 | 2.598 |
| III(MiCS) | **0.524** | 13.039 | 0.871 | 2.343 |
| IIG(PaRO) | **0.524** | 5.704 | 0.871 | 1.884 |
| IGG(PaRO) | 0.067 | 4.482 | 0.511 | 1.808 |
| GNG | 0.035 | 15.891 | 0.037 | 1.375 |
| GIG | 0.036 | 4.482 | 0.037 | 0.662 |
| GGG(ZeRO-3) | 0.024 | 3.26 | 0.036 | 0.586 |

Table 9: The Metric $(1/T)$ of training throughput and GPU memory of LLaMA-65B. $\Psi' = \Psi$, $\Psi' = \Psi/16$, and PEFT($\Psi' = 3\Psi/1000$) mean the different ratios of trainable parameters to model parameters. $-$ means out of GPU memory.

| Strategy ($p/g/os$) | $\Psi' = \Psi$ | | $\Psi' = \Psi/16$ | | $\Psi' = 3\Psi/1000$ | |
|---|---|---|---|---|---|---|
| | $1/T$ | Mem(GB) | $1/T$ | Mem(GB) | $1/T$ | Mem(GB) |
| NII | - | - | - | - | - | - |
| NIG(PaRO) | - | - | - | - | - | - |
| NGG(ZeRO-2) | - | - | - | - | - | - |
| INI | - | - | 0.091 | 28.376 | **0.098** | 15.77 |
| ING | - | - | 0.091 | 24.12 | **0.098** | 15.565 |
| III(MiCS) | - | - | **0.094** | 21.755 | **0.098** | 15.452 |
| IIG(PaRO) | **0.057** | 52.969 | **0.094** | 17.499 | **0.098** | 15.247 |
| IGG(PaRO) | 0.007 | 41.618 | 0.055 | 16.789 | 0.095 | 15.213 |
| GNG | - | - | 0.004 | 12.769 | 0.004 | 4.215 |
| GIG | 0.004 | 41.618 | 0.004 | 6.148 | 0.004 | 3.897 |
| GGG(ZeRO-3) | 0.003 | 30.268 | 0.004 | 5.439 | 0.004 | 3.863 |

The outcomes of the guidelines are presented in Table 8 and Table 9. Strategies highlighted in red indicate recommendations, distinguished by a higher throughput indicator $1/T$.

## A.4 Detail of Experiments

### A.4.1 Communication Performance of the Experimental Cluster

Table 10 presents the measured inter- and intra-group GPU-to-GPU communication throughput in the experimental environment.

Table 10: The Throughput of GPU to GPU communication.

| GPU to GPU | Size (Bytes) | Duration | Throughput (Gbps) |
|---|---|---|---|
| intra-node | 31.74 GB | 131.128 ms | 2079.138 |
| inter-node | 31.74 GB | 3033.485 ms | 89.875 |

### A.4.2 Results of the Experiments

Tables 11, 12, and 13 summarize the experimental results from Section 4.3, showcasing model configurations, throughput metrics, and GPU memory utilization for full-parameter, partial-parameter, and PEFT training scenarios. These tables offer a clear and comprehensive evaluation of the effectiveness of PaRO-DP.

Table 11: Configuration and Result of Experiments when full-parameters training ($\Psi' = \Psi$).

| Model Size | Strategy ($p/g/os$) | Config | Throughput | Peak Memory |
|---|---|---|---|---|
| 7B | GGG(ZeRO-3) | {"stage": 3} | 1.93 | 37.13 |
| | IGG(PaRO) | {"paro_strategy": "IGG"} | 2.84 | 44.76 |
| | IIG(PaRO) | {"paro_strategy": "IIG"} | 4.40 | 46.04 |
| | III(MiCS) | {"stage": 3, "mics_partition_size": 8} | 4.41 | 49.20 |
| | NGG(ZeRO-2) | {"stage": 2} | 2.79 | 52.58 |
| | NIG(PaRO) | {"paro_strategy": "NIG"} | 4.44 | 44.06 |
| | NII(PaRO) | {"paro_strategy": "NII"} | **4.65** | 51.98 |
| | ZeRO++ | {"stage": 3, "zero_hpz_partition_size": 8} | 2.43 | 39.76 |
| | FSDP-hz | HYBRID_SHARD_ZERO2 | 2.86 | 59.15 |
| 65B | GGG(ZeRO-3) | {"stage": 3} | 0.21 | 65.56 |
| | IGG(PaRO) | {"paro_strategy": "IGG"} | 0.29 | 76.12 |
| | IIG(PaRO) | {"paro_strategy": "IIG"} | **0.39** | 76.27 |
| | ZeRO++ | {"stage": 3, "zero_hpz_partition_size": 8} | 0.25 | 77.54 |

Table 12: Configuration and Result of Experiments when partial-parameters training ($\Psi' = \Psi/16$).

| Model Size | Strategy ($p/g/os$) | Config | Throughput | Peak Memory |
|---|---|---|---|---|
| 7B | GGG(ZeRO-3) | {"stage": 3} | 4.65 | 10.63 |
| | IGG(PaRO) | {"paro_strategy": "IGG"} | 9.41 | 11.97 |
| | IIG(PaRO) | {"paro_strategy": "IIG"} | 11.11 | 12.14 |
| | III(MiCS) | {"stage": 3, "mics_partition_size": 8} | 11.59 | 12.62 |
| | NGG(ZeRO-2) | {"stage": 2} | 11.94 | 18.86 |
| | NIG(PaRO) | {"paro_strategy": "NIG"} | 13.79 | 20.10 |
| | NII(PaRO) | {"paro_strategy": "NII"} | **14.46** | 19.66 |
| | ZeRO++ | {"stage": 3, "zero_hpz_partition_size": 8} | 4.79 | 12.34 |
| | FSDP-hz | HYBRID_SHARD_ZERO2 | 12.50 | 21.23 |
| 65B | GGG(ZeRO-3) | {"stage": 3} | 0.59 | 20.48 |
| | III(MiCS) | {"stage": 3, "mics_partition_size": 8} | 2.15 | 44.23 |
| | IIG(PaRO) | {"paro_strategy": "IIG"} | **2.16** | 39.48 |
| | ZeRO++ | {"stage": 3, "zero_hpz_partition_size": 8} | 0.60 | 34.72 |

Table 13: Configuration and Result of Experiments when PEFT ($\Psi' = 3\Psi/1000$).

| Model Size | Strategy ($p/g/os$) | Config | Throughput | Peak Memory |
|---|---|---|---|---|
| 65B | GGG(ZeRO-3) | {"stage": 3} | 1.20 | 29.54 |
| | ING(PaRO) | {"paro_strategy": "ING"} | **1.73** | 47.71 |
| | ZeRO++ | {"stage": 3, "zero_hpz_partition_size": 8} | 1.28 | 44.68 |

### A.4.3 Maximum Throughput via Dynamic Effective Batch Size

In Section 4.3, we standardized experimental conditions with a micro_batch_size of 4 and 10 accumulation_steps, setting an effective_batch_size of 40 per GPU for a fair comparison. This section presents the impact of dynamic effective batch size on training efficiency, specifically in the context of full-parameter training scenario within the A100 cluster. As displayed in the Table 14, the throughput of PaRO-NNI is 48.7% higher than that of ZeRO-1 using the same effective batch size. Additionally, the PaRO-IIG strategy achieves a throughput of 0.62, surpassing the GGG strategy's 0.47 by 32%, as shown in Table 15. This demonstrates the significant effectiveness of the IIG strategy in this scenario, which balances communication costs and memory usage, reducing inter-group communication and consequently improving overall training throughput.

Table 14: Throughput Comparison of LLaMA-7B on 32 GPUs when full-parameters training ($\Psi' = \Psi$). The global batch size in one global step is fixed to 17280 or effective batch size per GPU is fixed to 540. Note: MBS = micro_batch_size, AS = accumulation_steps, EBS = effective_batch_size

| Strategy ($p/g/os$) | Configuration (MBS, AS, EBS) of Single GPU | | |
|---|---|---|---|
| | (180, 3, 540) | (270, 2, 540) | (540, 1, 540) |
| NNG (ZeRO-1) | **5.908** | OOM | OOM |
| IIG | 5.493 | OOM | OOM |
| NII | 5.416 | **8.785** | OOM |

Table 15: Throughput Comparison of LLaMA-65B on 64 GPUs when full-parameters training ($\Psi' = \Psi$). Note: MBS = micro_batch_size, AS = accumulation_steps, EBS = effective_batch_size

| Strategy ($p/g/os$) | Configuration (MBS, AS, EBS) of Single GPU | | | | |
|---|---|---|---|---|---|
| | (16, 10, 160) | (20, 8, 160) | (32, 5, 160) | (40, 4, 160) | (48, -, -) |
| IIG (PaRO) | **0.62** | 0.57 | OOM | OOM | OOM |
| GGG (ZeRO-3) | 0.23 | 0.28 | 0.41 | **0.47** | OOM |

